# Maximum Margin Clustering

**Linli Xu**[*][†]  **James Neufeld**[†]  **Bryce Larson**[†]  **Dale Schuurmans**[†]

[*]University of Waterloo
[†]University of Alberta

## Abstract

We propose a new method for clustering based on finding maximum margin hyperplanes through data. By reformulating the problem in terms of the implied equivalence relation matrix, we can pose the problem as a convex integer program. Although this still yields a difficult computational problem, the hard-clustering constraints can be relaxed to a soft-clustering formulation which can be feasibly solved with a semidefinite program. Since our clustering technique only depends on the data through the kernel matrix, we can easily achieve nonlinear clusterings in the same manner as spectral clustering. Experimental results show that our maximum margin clustering technique often obtains more accurate results than conventional clustering methods. The real benefit of our approach, however, is that it leads naturally to a semi-supervised training method for support vector machines. By maximizing the margin simultaneously on labeled and unlabeled training data, we achieve state of the art performance by using a single, integrated learning principle.

## 1  Introduction

Clustering is one of the oldest forms of machine learning. Nevertheless, it has received a significant amount of renewed attention with the advent of nonlinear clustering methods based on kernels. Kernel based clustering methods continue to have a significant impact on recent work in machine learning [14, 13], computer vision [16], and bioinformatics [9].

Although many variations of kernel based clustering has been proposed in the literature, most of these techniques share a common "spectral clustering" framework that follows a generic recipe: one first builds the kernel ("affinity") matrix, normalizes the kernel, performs dimensionality reduction, and finally clusters (partitions) the data based on the resulting representation [17].

In this paper, our primary focus will be on the final partitioning step where the actual clustering occurs. Once the data has been preprocessed and a kernel matrix has been constructed (and its rank possibly reduced), many variants have been suggested in the literature for determining the final partitioning of the data. The predominant strategies include using k-means clustering [14], minimizing various forms of graph cut cost [13] (relaxations of which amount to clustering based on eigenvectors [17]), and finding strongly connected components in a Markov chain defined by the normalized kernel [4]. Some other recent alternatives are correlation clustering [12] and support vector clustering [1].

What we believe is missing from this previous work however, is a simple connection to

other types of machine learning, such as semisupervised and supervised learning. In fact, one of our motivations is to seek unifying machine learning principles that can be used to combine different types of learning problems in a common framework. For example, a useful goal for any clustering technique would be to find a way to integrate it seamlessly with a supervised learning technique, to obtain a principled form of semisupervised learning. A good example of this is [18], which proposes a general random field model based on a given kernel matrix. They then find a soft cluster assignment on unlabeled data that minimizes a joint loss with observed labels on supervised training data. Unfortunately, this technique actually *requires* labeled data to cluster the unlabeled data. Nevertheless, it is a useful approach.

Our goal in this paper is to investigate another standard machine learning principle—maximum margin classification—and modify it for clustering, with the goal of achieving a simple, unified way of solving a variety of problems, including clustering and semisupervised learning.

Although one might be skeptical that clustering based on large margin discriminants can perform well, we will see below that, combined with kernels, this strategy can often be more effective than conventional spectral clustering. Perhaps more significantly, it also immediately suggests a simple semisupervised training technique for support vector machines (SVMs) that appears to improve the state of the art.

The remainder of this paper is organized as follows. After establishing the preliminary ideas and notation in Section 2, we tackle the problem of computing a maximum margin clustering for a given kernel matrix in Section 3. Although it is not obvious that this problem can be solved efficiently, we show that the optimal clustering problem can in fact be formulated as a convex integer program. We then propose a relaxation of this problem which yields a semidefinite program that can be used to efficiently compute a soft clustering. Section 4 gives our experimental results for clustering. Then, in Section 5 we extend our approach to semisupervised learning by incorporating additional labeled training data in a seamless way. We then present experimental results for semisupervised learning in Section 6 and conclude.

## 2   Preliminaries

Since our main clustering idea is based on finding maximum margin separating hyperplanes, we first need to establish the background ideas from SVMs as well as establish the notation we will use.

For SVM training, we assume we are given labeled training examples $(\mathbf{x}^1, y^1), ..., (\mathbf{x}^N, y^N)$, where each example is assigned to one of two classes $y^i \in \{-1, +1\}$. The goal of an SVM of course is to find the linear discriminant $f_{\mathbf{w},b}(\mathbf{x}) = \mathbf{w}^\top \phi(\mathbf{x}) + b$ that maximizes the minimum misclassification margin

$$\gamma^* = \max_{\mathbf{w},b,\gamma} \gamma \quad \text{subject to} \quad y^i(\mathbf{w}^\top \phi(\mathbf{x}^i) + b) \geq \gamma, \forall_{i=1}^N, \|\mathbf{w}\|_2 = 1 \qquad (1)$$

Here the Euclidean normalization constraint on $\mathbf{w}$ ensures that the Euclidean distance between the data and the separating hyperplane (in $\phi(\mathbf{x})$ space) determined by $\mathbf{w}^*, b^*$ is maximized. It is easy to show that this same $\mathbf{w}^*, b^*$ is a solution to the quadratic program

$$\gamma^{*-2} = \min_{\mathbf{w},b} \|\mathbf{w}\|^2 \quad \text{subject to} \quad y^i(\mathbf{w}^\top \phi(\mathbf{x}^i) + b) \geq 1, \forall_{i=1}^N \qquad (2)$$

Importantly, the minimum value of this quadratic program, $\gamma^{*-2}$, is just the inverse square of the optimal solution value $\gamma^*$ to (1) [10].

To cope with potentially inseparable data, one normally introduces slack variables to reduce the dependence on noisy examples. This leads to the so called soft margin SVM (and its

dual) which is controlled by a tradeoff parameter $C$

$$
\begin{aligned}
\gamma^{*-2} &= \min_{\mathbf{w},b,\boldsymbol{\epsilon}} \|\mathbf{w}\|^2 + C\boldsymbol{\epsilon}^\top \mathbf{e} \quad \text{subject to} \quad y^i(\mathbf{w}^\top \boldsymbol{\phi}(\mathbf{x}^i) + b) \geq 1 - \epsilon_i, \forall_{i=1}^N,\ \boldsymbol{\epsilon} \geq 0 \\
&= \max_{\boldsymbol{\lambda}} 2\boldsymbol{\lambda}^\top \mathbf{e} - \langle K \circ \boldsymbol{\lambda}\boldsymbol{\lambda}^\top, \mathbf{yy}^\top \rangle \quad \text{subject to} \quad 0 \leq \boldsymbol{\lambda} \leq C,\ \boldsymbol{\lambda}^\top y = 0 \quad (3)
\end{aligned}
$$

The notation we use in this dual formulation requires some explanation, since we will use it below: Here $K$ denotes the $N \times N$ kernel matrix formed from the inner products of feature vectors $\Phi = [\boldsymbol{\phi}(\mathbf{x}^1), ..., \boldsymbol{\phi}(\mathbf{x}^N)]$ such that $K = \Phi^\top \Phi$. Thus $k_{ij} = \boldsymbol{\phi}(\mathbf{x}^i)^\top \boldsymbol{\phi}(\mathbf{x}^j)$. The vector $\mathbf{e}$ denotes the vector of all 1 entries. We let $A \circ B$ denote componentwise matrix multiplication, and let $\langle A, B \rangle = \sum_{ij} a_{ij} b_{ij}$. Note that (3) is derived from the standard dual SVM by using the fact that $\boldsymbol{\lambda}^\top (K \circ \mathbf{yy}^\top) \boldsymbol{\lambda} = \langle K \circ \mathbf{yy}^\top, \boldsymbol{\lambda}\boldsymbol{\lambda}^\top \rangle = \langle K \circ \boldsymbol{\lambda}\boldsymbol{\lambda}^\top, \mathbf{yy}^\top \rangle$.

To summarize: for *supervised* maximum margin training, one takes a given set of labeled training data $(\mathbf{x}^1, y^1), ..., (\mathbf{x}^N, y^N)$, forms the kernel matrix $K$ on data inputs, forms the kernel matrix $\mathbf{yy}^\top$ on target outputs, sets the slack parameter $C$, and solves the quadratic program (3) to obtain the dual solution $\boldsymbol{\lambda}^*$ and the inverse square maximum margin value $\gamma^{*-2}$. Once these are obtained, one can then recover a classifier directly from $\boldsymbol{\lambda}^*$ [15].

Of course, our main interest initially is not to find a large margin classifier given labels on the data, but instead to find a *labeling* that results in a large margin classifier.

## 3   Maximum margin clustering

The clustering principle we investigate is to find a labeling so that if one were to subsequently run an SVM, the margin obtained would be maximal over all possible labellings. That is, given data $\mathbf{x}^1, .., \mathbf{x}^N$, we wish to assign the data points to two classes $y^i \in \{-1, +1\}$ so that the separation between the two classes is as wide as possible.

Unsurprisingly, this is a hard computational problem. However, with some reformulation we can express it as a *convex* integer program, which suggests that there might be some hope of obtaining practical solutions. However, more usefully, we can relax the integer constraint to obtain a semidefinite program that yields soft cluster assignments which approximately maximize the margin. Therefore, one can obtain soft clusterings efficiently using widely available software. However, before proceeding with the main development, there are some preliminary issues we need to address.

First, we clearly need to impose some sort of constraint on the class balance, since otherwise one could simply assign all the data points to the same class and obtain an unbounded margin. A related issue is that we would also like to avoid the problem of separating a single outlier (or very small group of outliers) from the rest of the data. Thus, to mitigate these effects we will impose a constraint that the difference in class sizes be bounded. This will turn out to be a natural constraint for semisupervised learning and is very easy to enforce. Second, we would like the clustering to behave gracefully on noisy data where the classes may in fact overlap, so we adopt the soft margin formulation of the maximum margin criterion. Third, although it is indeed possible to extend our approach to the multiclass case [5], the extension is not simple and for ease of presentation we focus on simple two class clustering in this paper. Finally, there is a small technical complication that arises with one of the SVM parameters: It turns out that an unfortunate nonconvexity problem arises when we include the use of the offset $b$ in the underlying large margin classifier. We currently do not have a way to avoid this nonconvexity, and therefore we currently set $b = 0$ and therefore only consider homogeneous linear classifiers. The consequence of this restriction is that the constraint $\boldsymbol{\lambda}^\top y = 0$ is removed from the dual SVM quadratic program (3). Although it would seem like this is a harsh restriction, the negative effects are mitigated by centering the data at the origin, which can always be imposed. Nevertheless, dropping this

restriction remains an important question for future research. With these caveats in mind, we proceed to the main development.

We wish to solve for a labeling $\mathbf{y} \in \{-1, +1\}^N$ that leads to a maximum (soft) margin. Straightforwardly, one could attempt to tackle this optimization problem by directly formulating

$$\min_{\mathbf{y} \in \{-1,+1\}^N} \quad \gamma^{*-2}(\mathbf{y}) \quad \text{subject to} \quad -\ell \leq \mathbf{e}^\top \mathbf{y} \leq \ell \quad \text{where}$$

$$\gamma^{*-2}(\mathbf{y}) \quad = \quad \max_{\boldsymbol{\lambda}} 2\boldsymbol{\lambda}^\top \mathbf{e} - \langle K \circ \boldsymbol{\lambda}\boldsymbol{\lambda}^\top, \mathbf{y}\mathbf{y}^\top \rangle \quad \text{subject to} \quad 0 \leq \boldsymbol{\lambda} \leq C$$

Unfortunately, $\gamma^{*-2}(\mathbf{y})$ is not a convex function of $\mathbf{y}$, and this formulation does not lead to an effective algorithmic approach. In fact, to obtain an efficient technique for solving this problem we need two key insights.

The first key step is to re-express this optimization, not directly in terms of the cluster labels $\mathbf{y}$, but instead in terms of the label kernel matrix $M = \mathbf{y}\mathbf{y}^\top$. The main advantage of doing so is that the inverse soft margin $\gamma^{*-2}$ is in fact a convex function of $M$

$$\gamma^{*-2}(M) \quad = \quad \max_{\boldsymbol{\lambda}} 2\boldsymbol{\lambda}^\top \mathbf{e} - \langle K \circ \boldsymbol{\lambda}\boldsymbol{\lambda}^\top, M \rangle \quad \text{subject to} \quad 0 \leq \boldsymbol{\lambda} \leq C \qquad (4)$$

The convexity of $\gamma^{*-2}$ with respect to $M$ is easy to establish since this quantity is just a maximum over linear functions of $M$ [3]. This observation parallels one of the key insights of [10], here applied to $M$ instead of $K$.

Unfortunately, even though we can pose a convex objective, it does not allow us to immediately solve our problem because we still have to relate $M$ to $\mathbf{y}$, and $M = \mathbf{y}\mathbf{y}^\top$ is not a convex constraint. Thus, the main challenge is to find a way to constrain $M$ to ensure $M = \mathbf{y}\mathbf{y}^\top$ while respecting the class balance constraints $-\ell \leq \mathbf{e}^\top \mathbf{y} \leq \ell$. One obvious way to enforce $M = \mathbf{y}\mathbf{y}^\top$ would be to impose the constraint that $rank(M) = 1$, since combined with $M \in \{-1, +1\}^{N \times N}$ this forces $M$ to have a decomposition $\mathbf{y}\mathbf{y}^\top$ for some $\mathbf{y} \in \{-1, +1\}^N$. Unfortunately, $rank(M) = 1$ is not a convex constraint on $M$ [7].

Our second key idea is to realize that one can *indirectly* enforce the desired relationship $M = \mathbf{y}\mathbf{y}^\top$ by imposing a different set of linear constraints on $M$. To do so, notice that any such $M$ must encode an *equivalence relation* over the training points. That is, if $M = \mathbf{y}\mathbf{y}^\top$ for some $\mathbf{y} \in \{-1, +1\}^N$ then we must have

$$m_{ij} \quad = \quad \begin{cases} 1 & \text{if } y_i = y_j \\ -1 & \text{if } y_i \neq y_j \end{cases}$$

Therefore to enforce the constraint $M = \mathbf{y}\mathbf{y}^\top$ for $\mathbf{y} \in \{-1, +1\}^N$ it suffices to impose the set of constraints: (1) $M$ encodes an equivalence relation, namely that it is transitive, reflexive and symmetric; (2) $M$ has at most two equivalence classes; and (3) $M$ has at least two equivalence classes. Fortunately we can enforce each of these requirements by imposing a set of linear constraints on $M \in \{-1, +1\}^{N \times N}$ respectively:

$\mathcal{L}_1$: $m_{ii} = 1$; $m_{ij} = m_{ji}$; $m_{ik} \geq m_{ij} + m_{jk} - 1$; $\forall_{ijk}$

$\mathcal{L}_2$: $m_{jk} \geq -m_{ij} - m_{ik} - 1$; $\forall_{ijk}$

$\mathcal{L}_3$: $\sum_i m_{ij} \leq N - 2$; $\forall_j$

The result is that with only linear constraints on $M$ we can enforce the condition $M = \mathbf{y}\mathbf{y}^\top$.[1] Finally, we can enforce the class balance constraint $-\ell \leq \mathbf{e}^\top \mathbf{y} \leq \ell$ by imposing the additional set of linear constraints:

$\mathcal{L}_4$: $-\ell \leq \sum_i m_{ij} \leq \ell$; $\forall_j$

which obviates $\mathcal{L}_3$.

The combination of these two steps leads to our first main result: One can solve for a hard clustering $\mathbf{y}$ that maximizes the soft margin by solving a convex integer program. To accomplish this, one first solves for the equivalence relation matrix $M$ in

$$\min_{M \in \{-1,+1\}^{N \times N}} \max_{\boldsymbol{\lambda}} \; 2\boldsymbol{\lambda}^\top \mathbf{e} - \langle K \circ \boldsymbol{\lambda}\boldsymbol{\lambda}^\top, M \rangle \text{ subject to } 0 \leq \boldsymbol{\lambda} \leq C, \mathcal{L}_1, \mathcal{L}_2, \mathcal{L}_4 \qquad (5)$$

Then, from the solution $M^*$ recover the optimal cluster assignment $\mathbf{y}^*$ simply by setting $\mathbf{y}^*$ to any column vector in $M^*$.

Unfortunately, the formulation (5) is still not practical because convex integer programming is still a hard computational problem. Therefore, we are compelled to take one further step and relax the integer constraint on $M$ to obtain a convex optimization problem over a continuous parameter space

$$\min_{M \in [-1,+1]^{N \times N}} \max_{\boldsymbol{\lambda}} \; 2\boldsymbol{\lambda}^\top \mathbf{e} - \langle K \circ \boldsymbol{\lambda}\boldsymbol{\lambda}^\top, M \rangle \text{ subject to } 0 \leq \boldsymbol{\lambda} \leq C, \mathcal{L}_1, \mathcal{L}_2, \mathcal{L}_4, M \succeq 0 \quad (6)$$

This can be turned into an equivalent semidefinite program using essentially the same derivation as in [10], yielding

$$\min_{M,\delta,\boldsymbol{\mu},\boldsymbol{\nu}} \delta \quad \text{subject to} \quad \mathcal{L}_1, \mathcal{L}_2, \mathcal{L}_4, \boldsymbol{\mu} \geq 0, \boldsymbol{\nu} \geq 0, M \succeq 0 \qquad (7)$$

$$\begin{bmatrix} M \circ K & \mathbf{e} + \boldsymbol{\mu} - \boldsymbol{\nu} \\ (\mathbf{e} + \boldsymbol{\mu} - \boldsymbol{\nu})^\top & \delta - 2C\boldsymbol{\nu}^\top \mathbf{e} \end{bmatrix} \succeq 0$$

This gives us our second main result: To solve for a soft clustering $\mathbf{y}$ that approximately maximizes the soft margin, first solve the semidefinite program (7), and then from the solution matrix $M^*$ recover the soft cluster assignment $\mathbf{y}$ by setting $\mathbf{y} = \sqrt{\lambda_1}\mathbf{v}_1$, where $\lambda_1, \mathbf{v}_1$ are the maximum eigenvalue and corresponding eigenvector of $M^*$.[2]

## 4 Experimental results

We implemented the maximum margin clustering algorithm based on the semidefinite programming formulation (7), using the SeDuMi library, and tested it on various data sets.

In these experiments we compared the performance of our maximum margin clustering technique to the spectral clustering method of [14] as well as straightforward k-means clustering. Both maximum margin clustering and spectral clustering were run with the same radial basis function kernel and matching width parameters. In fact, in each case, we chose the best width parameter *for spectral clustering* by searching over a small set of five widths related to the scale of the problem. In addition, the slack parameter for maximum margin clustering was simply set to an arbitrary value.[3]

To assess clustering performance we first took a set of labeled data, removed the labels, ran the clustering algorithms, labeled each of the resulting clusters with the majority class according to the original training labels, and finally measured the number of misclassifications made by each clustering.

Our first experiments were conducted on the synthetic data sets depicted in Figure 1. Table 1 shows that for the first three sets of data (Gaussians, Circles, AI) maximum margin and spectral clustering obtained identical small error rates, which were in turn significantly

smaller than those obtained by k-means. However, maximum margin clustering demonstrates a substantial advantage on the fourth data set (Joined Circles) over both spectral and k-means clustering.

We also conducted clustering experiments on the real data sets, two of which are depicted in Figures 2 and 3: a database of images of handwritten digits of twos and threes (Figure 2), and a database of face images of two people (Figure 3). The last two columns of Table 1 show that maximum margin clustering obtains a slight advantage on the handwritten digits data, and a significant advantage on the faces data.

## 5  Semi-supervised learning

Although the clustering results are reasonable, we have an additional goal of adapting the maximum margin approach to semisupervised learning. In this case, we assume we are given a labeled training set $(\mathbf{x}^1, y^1), ..., (\mathbf{x}^n, y^n)$ as well as an unlabeled training set $\mathbf{x}^{n+1}, ..., \mathbf{x}^N$, and the goal is to combine the information in these two data sets to produce a more accurate classifier.

In the context of large margin classifiers, many techniques have been proposed for incorporating unlabeled data in an SVM, most of which are intuitively based on ensuring that large margins are also preserved on the unlabeled training data [8, 2], just as in our case. However, none of these previous proposals have formulated a convex optimization procedure that was guaranteed to directly maximize the margin, as we propose in Section 3.

For our procedure, extending the maximum margin clustering approach of Section 3 to semisupervised training is easy: We simply add constraints on the matrix $M$ to force it to respect the observed equivalence relations among the labeled training data. In addition, we impose the constraint that each unlabeled example belongs to the same class as at least one labeled training example. These conditions can be enforced with the simple set of additional linear constraints

$\mathcal{S}_1$:  $m_{ij} = y_i y_j$  for *labeled* examples $i, j \in \{1, ..., n\}$

$\mathcal{S}_2$:  $\sum_{i=1}^{n} m_{ij} \geq 2 - n$  for *unlabeled* examples $j \in \{n+1, ..., N\}$

Note that the observed training labels $y_i$ for $i \in \{1, ..., n\}$ are *constants*, and therefore the new constraints are still linear in the parameters of $M$ that are being optimized.

The resulting training procedure is similar to that of [6], with the addition of the constraints $\mathcal{L}_1 - \mathcal{L}_4, \mathcal{S}_2$ which enforce two classes and facilitate the ability to perform clustering on the unlabeled examples.

## 6  Experimental results

We tested our approach to semisupervised learning on various two class data sets from the UCI repository. We compared the performance of our technique to the semisupervised SVM technique of [8]. In each case, we evaluated the techniques transductively. That is, we split the data into a labeled and unlabeled part, held out the labels of the unlabeled portion, trained the semisupervised techniques, reclassified the unlabeled examples using the learned results, and measured the misclassification error on the held out labels.

Here we see that the maximum margin approach based on semidefinite programming can often outperform the approach of [8]. Table 2 shows that our maximum margin method is effective at exploiting unlabeled data to improve the prediction of held out labels. In every case, it significantly reduces the error of plain SVM, and obtains the best overall performance of the semisupervised learning techniques we have investigated.

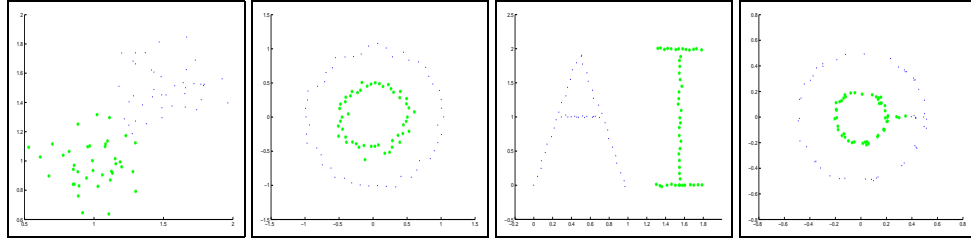

Figure 1: Four artificial data sets used in the clustering experiments. Each data set consists of eighty two-dimensional points. The points and stars show the two classes discovered by maximum margin clustering.

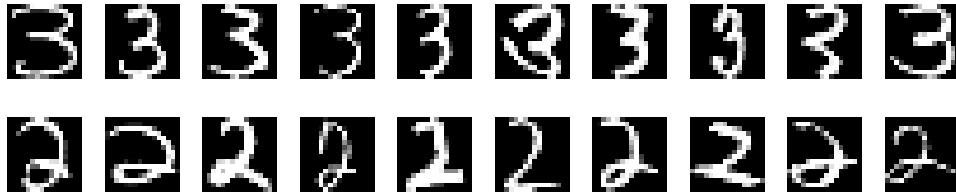

Figure 2: A sampling of the handwritten digits (twos and threes). Each row shows a random sampling of images from a cluster discovered by maximum margin clustering. Maximum margin made very few misclassifications on this data set, as shown in Table 1.

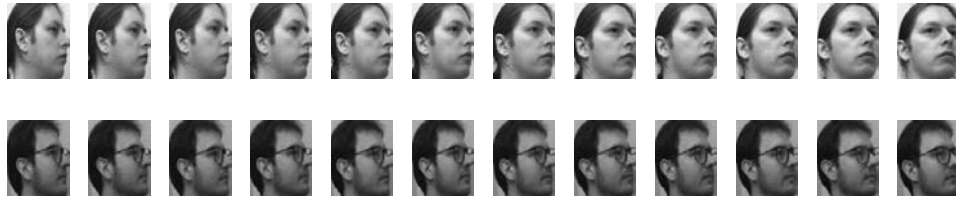

Figure 3: A sampling of the face data (two people). Each row shows a random sampling of images from a cluster discovered by maximum margin clustering. Maximum margin made no misclassifications on this data set, as shown in Table 1.

|  | Gaussians | Circles | A I | Joined Circles | Digits | Faces |
|---|---|---|---|---|---|---|
| Maximum Margin | 1.25 | 0 | 0 | 1 | 3 | 0 |
| Spectral Clustering | 1.25 | 0 | 0 | 24 | 6 | 16.7 |
| K-means | 5 | 50 | 38.5 | 50 | 7 | 24.4 |

Table 1: Percentage misclassification errors of the various clustering algorithms on the various data sets.

|  | HWD 1-7 | HWD 2-3 | UCI Austra. | UCI Flare | UCI Vote | UCI Diabet. |
|---|---|---|---|---|---|---|
| Max Marg | 3.3 | 4.7 | 32 | 34 | 14 | 35.55 |
| Spec Clust | 4.2 | 6.4 | 48.7 | 40.7 | 13.8 | 44.67 |
| TSVM | 4.6 | 5.4 | 38.7 | 33.3 | 17.5 | 35.89 |
| SVM | 4.5 | 10.9 | 37.5 | 37 | 20.4 | 39.44 |

Table 2: Percentage misclassification errors of the various semisupervised learning algorithms on the various data sets. SVM uses no unlabeled data. TSVM is due to [8].

# 7 Conclusion

We have proposed a simple, unified principle for clustering and semisupervised learning based on the maximum margin principle popularized by supervised SVMs. Interestingly, this criterion can be approximately optimized using an efficient semidefinite programming formulation. The results on both clustering and semisupervised learning are competitive with, and sometimes exceed the state of the art. Overall, margin maximization appears to be an effective way to achieve a unified approach to these different learning problems.

For future work we plan to address the restrictions of the current method, including the ommission of an offset $b$ and the restriction to two class problems. We note that a multiclass extension to our approach is possible, but it is complicated by the fact that it cannot be conveniently based on the standard multiclass SVM formulation of [5]

### Acknowledgements

Research supported by the Alberta Ingenuity Centre for Machine Learning, NSERC, MITACS, IRIS and the Canada Research Chairs program.

## Footnotes

[1]Interestingly, for $M \in \{-1, +1\}^{N \times N}$ the first two sets of linear constraints can be replaced by the compact set of convex constraints $diag(M) = \mathbf{e}$, $M \succeq 0$ [7, 11]. However, when we relax the integer constraint below, this equivalence is no longer true and we realize some benefit in keeping the linear equivalence relation constraints.

[2]One could also employ randomized rounding to choose a hard class assignment $\mathbf{y}$.

[3]It turns out that the slack parameter $C$ did not have a significant effect on any of our preliminary investigations, so we just set it to $C = 100$ for all of the experiments reported here.

## References

[1] A. Ben-Hur, D. Horn, H. Siegelman, and V. Vapnik. Support vector clustering. In *Journal of Machine Learning Research 2 (2001)*, 2001.

[2] K. Bennett and A. Demiriz. Semi-supervised support vector machines. In *Advances in Neural Information Processing Systems 11 (NIPS-98)*, 1998.

[3] S. Boyd and L. Vandenberghe. *Convex Optimization*. Cambridge U. Press, 2004.

[4] Chakra Chennubhotla and Allan Jepson. Eigencuts: Half-lives of eigenflows for spectral clustering. In *In Advances in Neural Information Processing Systems, 2002*, 2002.

[5] K. Crammer and Y. Singer. On the algorithmic interpretation of multiclass kernel-based vector machines. *Journal of Machine Learning Research*, 2, 2001.

[6] T. De Bie and N. Cristianini. Convex methods for transduction. In *Advances in Neural Information Processing Systems 16 (NIPS-03)*, 2003.

[7] C. Helmberg. Semidefinite programming for combinatorial optimization. Technical Report ZIB-Report ZR-00-34, Konrad-Zuse-Zentrum Berlin, 2000.

[8] T. Joachims. Transductive inference for text classification using support vector machines. In *International Conference on Machine Learning (ICML-99)*, 1999.

[9] Y. Kluger, R. Basri, J. Chang, and M. Gerstein. Spectral biclustering of microarray cancer data: co-clustering genes and conditions. *Genome Research*, 13, 2003.

[10] G. Lanckriet, N. Cristianini, P. Bartlett, L Ghaoui, and M. Jordan. Learning the kernel matrix with semidefinite programming. *Journal of Machine Learning Research*, 5, 2004.

[11] M. Laurent and S. Poljak. On a positive semidefinite relaxation of the cut polytope. *Linear Algebra and its Applications*, 223/224, 1995.

[12] S. Chawla N. Bansal, A. Blum. Correlation clustering. In *Conference on Foundations of Computer Science (FOCS-02)*, 2002.

[13] J. Kandola N. Cristianini, J. Shawe-Taylor. Spectral kernel methods for clustering. In *In Advances in Neural Information Processing System, 2001*, 2001.

[14] A. Ng, M. Jordan, and Y Weiss. On spectral clustering: analysis and an algorithm. In *Advances in Neural Information Processing Systems 14 (NIPS-01)*, 2001.

[15] B. Schoelkopf and A. Smola. *Learning with Kernels: Support Vector Machines, Regularization, Optimization, and Beyond*. MIT Press, 2002.

[16] J. Shi and J. Malik. Normalized cuts and image segmentation. *IEEE Trans PAMI*, 22(8), 2000.

[17] Y. Weiss. Segmentation using eigenvectors: a unifying view. In *International Conference on Computer Vision (ICCV-99)*, 1999.

[18] X. Zhu, Z. Ghahramani, and J. Lafferty. Semi-supervised learning using gaussian fields and harmonic functions. In *International Conference on Machine Learning (ICML-03)*, 2003.
